# SPEAKER RECOGNITION USING NEURAL TREE NETWORKS

**Kevin R. Farrell and Richard J. Mammone**
CAIP Center, Rutgers University
Core Building, Frelinghuysen Road
Piscataway, New Jersey 08855

## Abstract

A new classifier is presented for text-independent speaker recognition. The new classifier is called the modified neural tree network (MNTN). The NTN is a hierarchical classifier that combines the properties of decision trees and feed-forward neural networks. The MNTN differs from the standard NTN in that a new learning rule based on *discriminant learning* is used, which minimizes the classification error as opposed to a norm of the approximation error. The MNTN also uses leaf probability measures in addition to the class labels. The MNTN is evaluated for several speaker identification experiments and is compared to multilayer perceptrons (MLPs), decision trees, and vector quantization (VQ) classifiers. The VQ classifier and MNTN demonstrate comparable performance and perform significantly better than the other classifiers for this task. Additionally, the MNTN provides a logarithmic saving in retrieval time over that of the VQ classifier. The MNTN and VQ classifiers are also compared for several speaker verification experiments where the MNTN is found to outperform the VQ classifier.

## 1 INTRODUCTION

Automatic speaker recognition consists of having a machine recognize a person based on his or her voice. Automatic speaker recognition is comprised of two categories: speaker identification and speaker verification. The objective of speaker identification is to identify a person within a fixed population based on a test utterance from that person. This is contrasted to speaker verification where the objective is to verify a person's claimed identity based on the test utterance.

Speaker recognition systems can be either text dependent or text independent. Text-dependent speaker recognition systems require that the speaker utter a specific phrase or a given password. Text-independent speaker identification systems identify the speaker regardless of the utterance. This paper focuses on text-independent speaker identification and speaker verification tasks.

A new classifier is introduced and evaluated for speaker recognition. The new classifier is the modified neural tree network (MNTN). The MNTN incorporates modifications to the learning rule of the original NTN [1] and also uses leaf probability measures in addition to the class labels. Also, vector quantization (VQ) classifiers, multilayer perceptrons (MLPs), and decision trees are evaluated for comparison.

This paper is organized as follows. Section 2 reviews the neural tree network and discusses the modifications. Section 3 discusses the feature extraction and classification phases used here for text-independent speaker recognition. Section 4 describes the database used and provides the experimental results. The summary and conclusions of the paper are given in Section 5.

## 2   THE MODIFIED NEURAL TREE NETWORK

The NTN [1] is a hierarchical classifier that uses a tree architecture to implement a sequential linear decision strategy. Each node at every level of the NTN divides the input training vectors into a number of exclusive subsets of this data. The leaf nodes of the NTN partition the feature space into homogeneous subsets, i.e., a single class at each leaf node. The NTN is recursively trained as follows. Given a set of training data at a particular node, if all data within that node belongs to the same class, the node becomes a leaf. Otherwise, the data is split into several subsets, which become the children of this node. This procedure is repeated until all the data is completely uniform at the leaf nodes.

For each node the NTN computes the inner product of a weight vector $w$ and an input feature vector $x$, which should be approximately equal to the the output label $y \in \{0, 1\}$. Traditional learning algorithms minimize a norm of the error $\epsilon = (y- < w, x >)$, such as the $L_2$ or $L_1$ norm. The splitting algorithm of the modified NTN is based on discriminant learning [2]. Discriminant learning uses a cost function that minimizes the classification error.

For an M class NTN, the discriminant learning approach first defines a misclassification measure $d(x)$ as [2]:

$$d(x) = - < w_i, x > + \left\{ \frac{1}{M-1} \sum_{j \neq i} (< w_j, x >)^n \right\}^{\frac{1}{n}},  \quad (1)$$

where $n$ is a predetermined smoothing constant. If $x$ belongs to class $i$, then $d(x)$ will be negative, and if $x$ does not belong to class $i$, $d(x)$ will be positive. The misclassification measure $d(x)$ is then applied to a sigmoid to yield:

$$g[d(x)] = \frac{1}{1 + e^{-d(x)}}.  \quad (2)$$

The cost function in equation (2) is approximately zero for correct classifications and one for misclassifications. Hence, minimizing this cost function will tend to

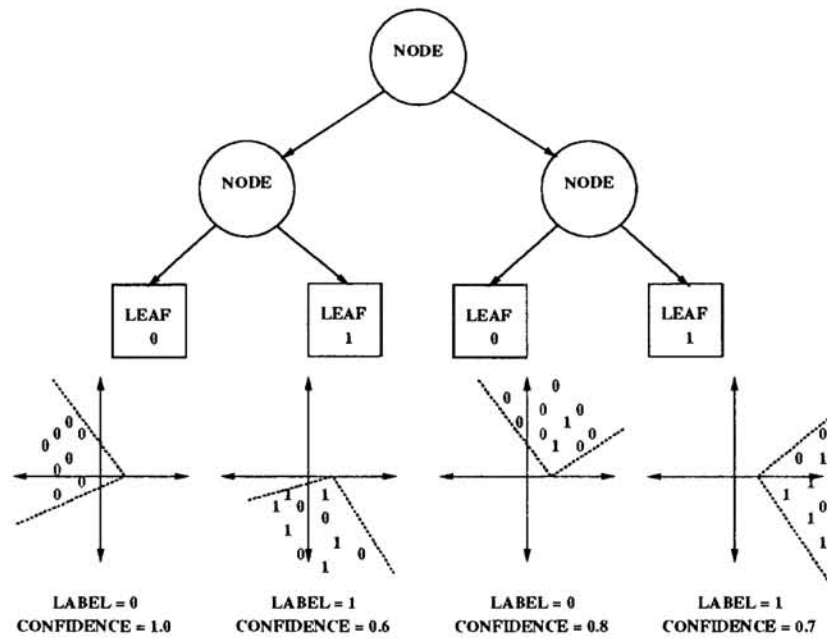

Figure 1: Forward Pruning and Confidence Measures

minimize the classification error. It is noted that for binary NTNs, the weight updates obtained by the discriminant learning approach and the $L_1$ norm of the error are equivalent [3].

The NTN training algorithm described above constructs a tree having 100% performance on the training set. However, an NTN trained to this level may not have optimal generalization due to overtraining. The generalization can be improved by reducing the number of nodes in a tree, which is known as *pruning*. A technique known as *backward* pruning was recently proposed [1] for the NTN. Given a fully grown NTN, i.e., 100% performance on the training set, the backward pruning method uses a Lagrangian cost function to minimize the classification error and the number of leaves in the tree. The method used here prunes the tree during its growth, hence it is called *forward* pruning.

The forward pruning algorithm consists of simply truncating the growth of the tree beyond a certain level. For the leaves at the truncated level, a vote is taken and the leaf is assigned the label of the majority. In addition to a label, the leaf is also assigned a confidence. The confidence is computed as the ratio of the number of elements for the vote winner to the total number of elements. The confidence provides a measure of confusion for the different regions of feature space. The concept of forward pruning is illustrated in Figure 1.

# 3    FEATURE EXTRACTION AND CLASSIFICATION

The process of feature extraction consists of obtaining characteristic parameters of a signal to be used to classify the signal. The extraction of salient features is a key step in solving any pattern recognition problem. For speaker recognition, the features extracted from a speech signal should be invariant with regard to the desired

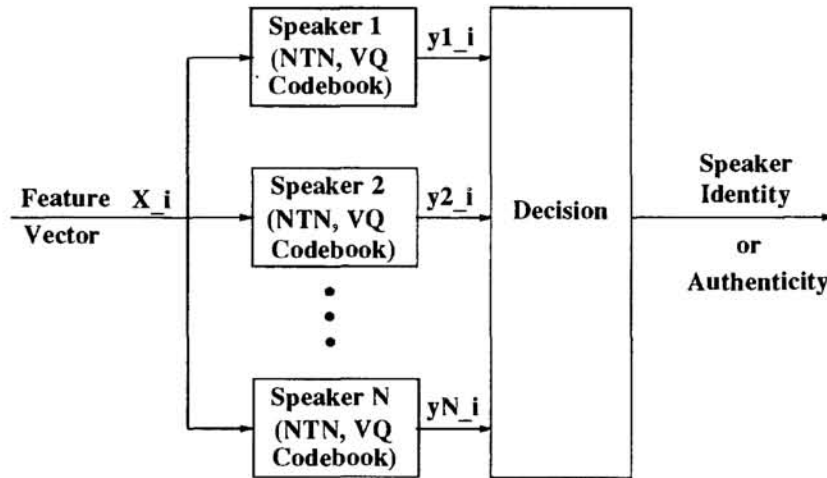

Figure 2: Classifier Structure for Speaker Recognition

speaker while exhibiting a large distance to any imposter. Cepstral coefficients are commonly used for speaker recognition [4] and shall be considered here to evaluate the classifiers.

The classification stage of text-independent speaker recognition is typically implemented by modeling each speaker with an individual classifier. The classifier structure for speaker recognition is illustrated in Figure 2. Given a specific feature vector, each speaker model associates a number corresponding to the degree of match with that speaker. The stream of numbers obtained for a set of feature vectors can be used to obtain a likelihood score for each speaker model. For speaker identification, the feature vectors for the test utterance are applied to all speaker models and the corresponding likelihood scores are computed. The speaker is selected as having the largest score. For speaker verification, the feature vectors are applied only to the speaker model for the speaker to be verified. If the likelihood score exceeds a threshold, the speaker is verified or else is rejected.

The classifiers for the individual speaker models are trained using either supervised or unsupervised training methods. For supervised training methods the classifier for each speaker model is presented with the data for all speakers. Here, the extracted feature vectors for that speaker are labeled as "one" and the extracted feature vectors for everyone else are labeled as "zero". The supervised classifiers considered here are the multilayer perceptron (MLP), decision trees, and modified neural tree network (MNTN). For unsupervised training methods each speaker model is presented with only the extracted feature vectors for that speaker. This data can then be clustered to determine a set of centroids that are representative of that speaker. The unsupervised classifiers evaluated here are the full-search and tree-structure vector quantization classifiers, henceforth denoted as FSVQ and TSVQ. Speaker models based on supervised training capture the differences of that speaker to other speakers, whereas models based on unsupervised training use a similarity measure.

Specifically, a trained NTN can be applied to speaker recognition as follows. Given a sequence of feature vectors $x$ from the test utterance and a trained NTN for

speaker $S_i$, the corresponding speaker score is found as the "hit" ratio:

$$P_{NTN}(x|S_i) = \frac{M}{N+M}.$$ (3)

Here, $M$ is the number of vectors classified as "one" and $N$ is the number of vectors classified as "zero". The modified NTN computes a hit ratio weighed by the confidence scores:

$$P_{MNTN}(x|S_i) = \frac{\sum_{j=1}^{M} c_j^1}{\sum_{j=1}^{N} c_j^0 + \sum_{j=1}^{M} c_j^1},$$ (4)

where $c^1$ and $c^0$ are the confidence scores for the speaker and antispeaker, respectively. These scores can be used for decisions regarding identification or verification.

## 4   EXPERIMENTAL RESULTS

### 4.1   Database

The database considered for the speaker identification and verification experiments is a subset of the DARPA TIMIT database. This set represents 38 speakers of the same (New England) dialect. The preprocessing of the TIMIT speech data consists of several steps. First, the speech is downsampled from 16KHz to 8 KHz sampling frequency. The downsampling is performed to obtain a toll quality signal. The speech data is processed by a silence removing algorithm followed by the application of a pre-emphasis filter $H(z) = 1 - 0.95z^{-1}$. A 30 ms Hamming window is applied to the speech every 10 ms. A twelfth order linear predictive (LP) analysis is performed for each speech frame. The features consist of the twelve cepstral coefficients derived from this LP polynomial.

There are 10 utterances for each speaker in the selected set. Five of the utterances are concatenated and used for training. The remaining five sentences are used individually for testing. The duration of the training data ranges from 7 to 13 seconds per speaker and the duration of each test utterance ranges from 0.7 to 3.2 seconds.

### 4.2   Speaker Identification

The first experiment is for closed set speaker identification using 10 and 20 speakers from the TIMIT New England dialect. The identification is closed set in that the speaker is assumed to be one of the 10 or 20 speakers, i.e., no "none of the above" option. The NTN, MLP [5], and VQ [4] classifiers are each evaluated on this data in addition to the ID3 [6], C4 [7], CART [8], and Bayesian [9] decision trees. The VQ classifier is trained using a K-means algorithm and tested for codebook sizes varying from 16 to 128 centroids. The MNTN used here is pruned at levels ranging from the fourth through seventh. The MLP is trained using the backpropagation algorithm [10] for architectures having 16, 32, and 64 hidden nodes (within one hidden layer). The results are summarized in Table 1. The * denotes that the CART tree could not be grown for the 20 speaker experiment due to memory limitations.

| Classifier | 10 speakers | 20 speakers |
|---|---|---|
| ID3 | 88% | 79% |
| CART | 76% | * |
| C4 | 84% | 73% |
| Bayes | 92% | 83% |
| MLP (16/32/64) | 90/90/94% | 90/82/85% |
| NTN (4/5/6/7) | 66/84/92/92% | 67/76/82/91% |
| MNTN (4/5/6/7) | 88/90/94/98% | 75/87/93/96% |
| TSVQ (16/32/64/128) | 92/92/96/94% | 83/90/90/88% |
| FSVQ (16/32/64/128) | 98/98/98/98% | 90/92/95/96% |

Table 1: Speaker Identification Experiments

## 4.3  Speaker Verification

The FSVQ classifier and MNTN are evaluated next for speaker verification. The first speaker verification experiment consists of 10 speakers and 10 imposters (i.e., people not used in the training set). The second speaker verification experiment uses 20 enrolled speakers and 18 imposters. The MNTN is pruned at the seventh level (128 leaves) and the FSVQ classifier has a codebook size of 128 entries.

Speaker verification performance can be enhanced by using a technique known as *cohort* normalization [11]. Traditional verification systems accept a speaker if:

$$p(X|I) > T(I), \qquad (5)$$

where $p(X|I)$ is the likelihood that the sequence of feature vectors $X$ was generated by speaker $I$ and $T(I)$ is the corresponding likelihood threshold. Instead of using the fixed threshold criteria in equation (5), an adaptive threshold can be used via the likelihood measure:

$$\frac{P(X|I)}{P(X|\bar{I})} > T(I). \qquad (6)$$

Here, the speaker score is first normalized by the probability that the feature vectors $X$ were generated by a speaker other than $I$. The likelihood $p(X|\bar{I})$ can be estimated with the scores of the speaker models that are closest to $I$, denoted as $I$'s *cohorts* [11]. This estimate can consist of a maximum, minimum, average, etc., depending on the classifier used.

The threshold for the VQ and MNTN likelihood scores are varied from the point of 0% false acceptance to 0% false rejection to yield the operating curves shown in Figures 3 and 4 for the 10 and 20 speaker populations, respectively. Note that all operating curves presented in this section for speaker verification represent the posterior performance of the classifiers, given the speaker and imposter scores. Here it can be seen that the MNTN and VQ classifiers are both improved by the cohort normalized scores. The equal error rates for the MNTN and VQ classifier are summarized in Table 2.

For both experiments (10 and 20 speakers), the MNTN provides better performance than the VQ classifier, both with and without cohort normalization, for most of the operating curve.

|  | VQ (w/o cohort) | VQ (w/ cohort) | MNTN (w/o cohort) | MNTN (w/ cohort) |
|---|---|---|---|---|
| 10 Speakers | 3.6% | 2.2% | 2.4% | 1.8% |
| 20 Speakers | 4.6% | 2.0% | 2.8% | 1.9% |

Table 2: Equal error rates for speaker verification

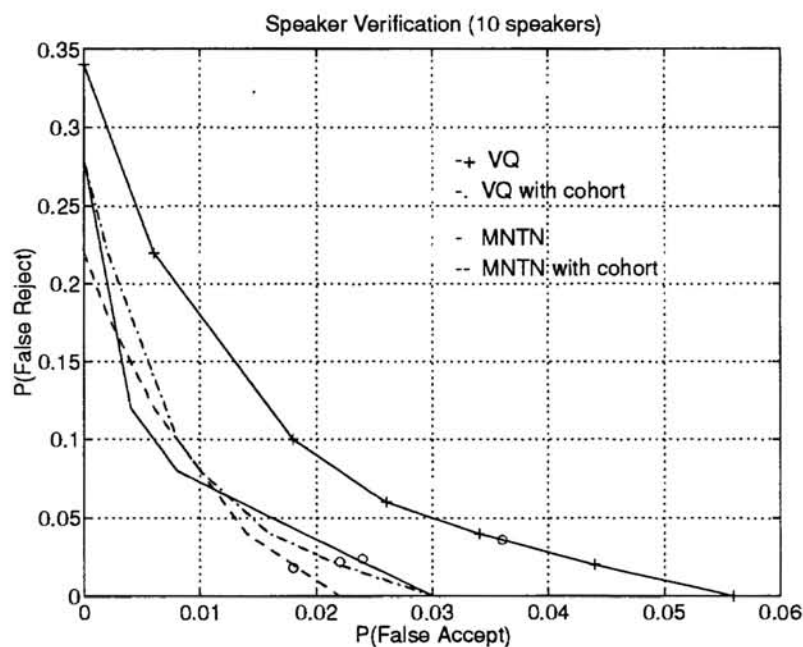

Figure 3: Speaker Verification (10 Speakers)

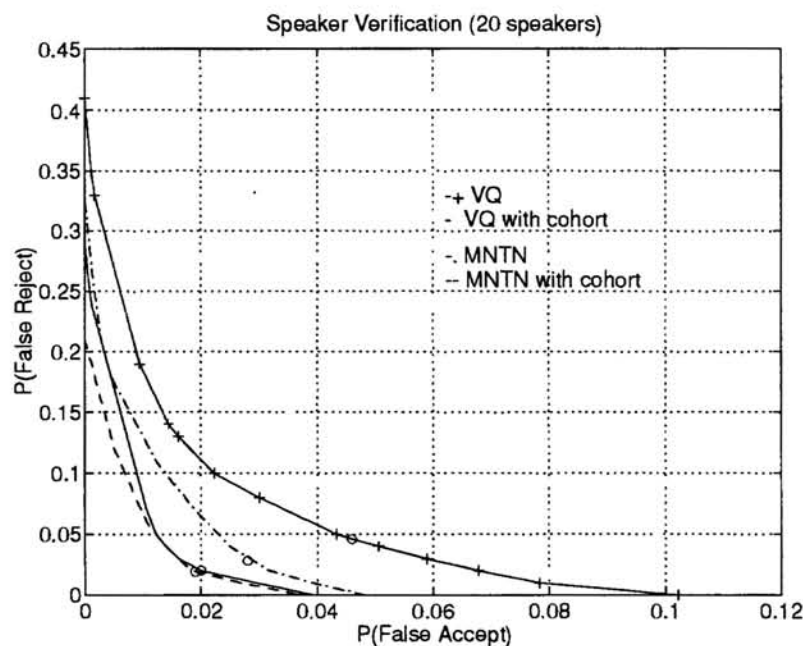

Figure 4: Speaker Verification (20 Speakers)

## 5    CONCLUSION

A new classifier called the modified NTN is examined for text-independent speaker recognition. The performance of the MNTN is evaluated for several speaker recognition experiments using various sized speaker sets from a 38 speaker corpus. The features used to evaluate the classifiers are the LP-derived cepstrum. The MNTN is compared to full-search and tree-structured VQ classifiers, multi-layer perceptrons, and decision trees. The FSVQ and MNTN classifiers both demonstrate equivalent performance for the speaker identification experiments and outperform the other classifiers. For speaker verification, the MNTN consistently outperforms the FSVQ classifier. In addition to performance advantages for speaker verification, the MNTN also demonstrates a logarithmic saving in retrieval time over that of the FSVQ classifier. This computational advantage can be obtained by using TSVQ, although TSVQ will reduce the performance with respect to FSVQ.

## 6    ACKNOWLEDGEMENTS

The authors gratefully acknowledge the support of Rome Laboratories, Contract No. F30602-91-C-0120. The decision tree simulations utilized the IND package developed by W. Buntine of NASA.

## References

[1] A. Sankar and R.J. Mammone. Growing and pruning neural tree networks. *IEEE Transactions on Computers*, C-42:221–229, March 1993.

[2] S. Katagiri, B.H Juang, and A. Biem. Discriminative feature extraction. In *Artificial Neural Networks for Speech and Vision Processing, edited by R.J. Mammone*. Chapman and Hall, 1993.

[3] K.R. Farrell. *Speaker Recognition Using the Modified Neural Tree Network*. PhD thesis, Rutgers University, Oct. 1993.

[4] F.K. Soong, A.E. Rosenberg, L.R. Rabiner, and B.H. Juang. A vector quantization approach to speaker recognition. In *Proceedings ICASSP*, 1985.

[5] J. Oglesby and J.S. Mason. Optimization of neural models for speaker identification. In *Proceedings ICASSP*, pages 261–264, 1990.

[6] J. Quinlan. Induction of decision trees. *Machine Learning*, 1:81–106, 1986.

[7] J. Quinlan. *Simplifying decision trees in Knowledge Acquisition for Knowledge-Based Systems, by G. Gaines and J. Boose*. Academic Press, London, 1988.

[8] L. Breiman, J.H. Friedman, R.A. Olshen, and C.J. Stone. *Classification and Regression Trees*. Wadsworth international group, Belmont, CA, 1984.

[9] W. Buntine. Learning classification trees. *Statistics and Computing*, 2:63–73, 1992.

[10] D.E. Rumelhart and J.L. McClelland. *Parallel Distributed Processing*. MIT Cambridge Press, Cambridge, Ma, 1986.

[11] A.E. Rosenberg, J. Delong, C.H. Lee, B.H. Juang, and F.K. Soong. The use of cohort normalized scores for speaker recognition. In *Proc. ICSLP*, Oct. 1992.